# Online Classification for Complex Problems Using Simultaneous Projections

**Yonatan Amit**[1]    **Shai Shalev-Shwartz**[1]    **Yoram Singer**[1,2]

[1] School of Computer Sci. & Eng., The Hebrew University, Jerusalem 91904, Israel
[2] Google Inc. 1600 Amphitheatre Pkwy, Mountain View, CA 94043, USA
`{mitmit,shais,singer}@cs.huji.ac.il`

## Abstract

We describe and analyze an algorithmic framework for online classification where each online trial consists of *multiple* prediction tasks that are tied together. We tackle the problem of updating the online hypothesis by defining a projection problem in which each prediction task corresponds to a single linear constraint. These constraints are tied together through a single slack parameter. We then introduce a general method for approximately solving the problem by projecting *simultaneously* and independently on each constraint which corresponds to a prediction sub-problem, and then averaging the individual solutions. We show that this approach constitutes a feasible, albeit not necessarily optimal, solution for the original projection problem. We derive concrete simultaneous projection schemes and analyze them in the mistake bound model. We demonstrate the power of the proposed algorithm in experiments with online multiclass text categorization. Our experiments indicate that a combination of class-dependent features with the simultaneous projection method outperforms previously studied algorithms.

## 1  Introduction

In this paper we discuss and analyze a framework for devising efficient online learning algorithms for complex prediction problems such as multiclass categorization. In the settings we cover, a complex prediction problem is cast as the task of simultaneously coping with multiple simplified sub-problems which are nonetheless tied together. For example, in multiclass categorization, the task is to predict a *single* label out of $k$ possible outcomes. Our simultaneous projection approach is based on the fact that we can retrospectively (after making a prediction) cast the problem as the task of making $k-1$ binary decisions each of which involves the correct label and one of the competing labels. The performance of the $k-1$ predictions is measured through a single loss. Our approach stands in contrast to previously studied methods which can be roughly be partitioned into three paradigms. The first and probably the simplest previously studied approach is to break the problem into multiple *decoupled* problems that are solved *independently*. Such an approach was used for instance by Weston and Watkins [1] for batch learning of multiclass support vector machines. The simplicity of this approach also underscores its deficiency as it is detached from the original loss of the complex decision problem. The second approach maintains the original structure of the problem but focuses on a single, worst performing, derived sub-problem (see for instance [2]). While this approach adheres with the original structure of the problem, the resulting update mechanism is by construction sub-optimal as it oversees almost all of the constraints imposed by the complex prediction problem. (See also [6] for analysis and explanation of the sub-optimality of this approach.) The third approach for dealing with complex problems is to tailor a specific efficient solution for the problem on hand. While this approach yielded efficient learning algorithms for multiclass categorization problems [2] and aesthetic solutions for structured output problems [3, 4], devising these algorithms required dedicated efforts. Moreover, tailored solutions typically impose rather restrictive assumptions on the representation of the data in order to yield efficient algorithmic solutions.

In contrast to previously studied approaches, we propose a simple, general, and efficient framework for online learning of a wide variety of complex problems. We do so by casting the online update task as an optimization problem in which the newly devised hypothesis is required to be similar to the current hypothesis while attaining a small loss on *multiple* binary prediction problems. Casting the online learning task as a sequence of instantaneous optimization problems was first suggested and analyzed by Kivinen and Warmuth [12] for binary classification and regression problems. In our optimization-based approach, the complex decision problem is cast as an optimization problem that consists of *multiple* linear constraints each of which represents a simplified sub-problem. These constraints are tied through a *single* slack variable whose role is to asses the *overall* prediction quality for the complex problem. We describe and analyze a family of two-phase algorithms. In the first phase, the algorithms solve *simultaneously* multiple sub-problems. Each sub-problem distills to an optimization problem with a *single* linear constraint from the original multiple-constraints problem. The simple structure of each single-constraint problem results in an analytical solution which is efficiently computable. In the second phase, the algorithms take a convex combination of the independent solutions to obtain a solution for the multiple-constraints problem. The end result is an approach whose time complexity and mistake bounds are equivalent to approaches which solely deal with the worst-violating constraint [9]. In practice, though, the performance of the simultaneous projection framework is much better than single-constraint update schemes.

## 2 Problem Setting

In this section we introduce the notation used throughout the paper and formally describe our problem setting. We denote vectors by lower case bold face letters (e.g. $\mathbf{x}$ and $\boldsymbol{\omega}$) where the $j$'th element of $\mathbf{x}$ is denoted by $x_j$. We denote matrices by upper case bold face letters (e.g. $\mathbf{X}$), where the $j$'th row of $\mathbf{X}$ is denoted by $\mathbf{x}_j$. The set of integers $\{1, \ldots, k\}$ is denoted by $[k]$. Finally, we use the hinge function $[a]_+ = \max\{0, a\}$.

Online learning is performed in a sequence of trials. At trial $t$ the algorithm receives a matrix $\mathbf{X}_t$ of size $k_t \times n$, where each row of $\mathbf{X}_t$ is an instance, and is required to make a prediction on the label associated with each instance. We denote the vector of predicted labels by $\hat{\mathbf{y}}^t$. We allow $\hat{y}_j^t$ to take any value in $\mathbb{R}$, where the actual label being predicted is $\text{sign}(\hat{y}_j^t)$ and $|\hat{y}_j^t|$ is the confidence in the prediction. After making a prediction $\hat{\mathbf{y}}^t$ the algorithm receives the correct labels $\mathbf{y}^t$ where $y_j^t \in \{-1, 1\}$ for all $j \in [k_t]$. In this paper we assume that the predictions in each trial are formed by calculating the inner product between a weight vector $\boldsymbol{\omega}^t \in \mathbb{R}^n$ with each instance in $\mathbf{X}^t$, thus $\hat{\mathbf{y}}^t = \mathbf{X}^t \boldsymbol{\omega}^t$. Our goal is to *perfectly* predict the entire vector $\mathbf{y}^t$. We thus say that the vector $\hat{\mathbf{y}}^t$ was *imperfectly* predicted if there exists an outcome $j$ such that $y_j^t \neq \text{sign}(\hat{y}_j^t)$. That is, we suffer a unit loss on trial $t$ if there exists $j$, such that $\text{sign}(\hat{y}_j^t) \neq y_j^t$. Directly minimizing this combinatorial error is a computationally difficult task. Therefore, we use an adaptation of the *hinge-loss*, defined $\ell(\hat{\mathbf{y}}^t, \mathbf{y}^t) = \max_{j \in [k_t]} \left[1 - y_j^t \hat{y}_j^t\right]_+$, as a proxy for the combinatorial error. The quantity $y_j^t \hat{y}_j^t$ is often referred to as the (signed) *margin* of the prediction and ties the correctness and the confidence in the prediction. We use $\ell(\boldsymbol{\omega}; (\mathbf{X}^t, \mathbf{y}^t))$ to denote $\ell(\hat{\mathbf{y}}^t, \mathbf{y}^t)$ where $\hat{\mathbf{y}}^t = \mathbf{X}^t \boldsymbol{\omega}$. We also denote the set of instances whose labels were predicted incorrectly by $\mathcal{M}^t = \{j \,|\, \text{sign}(\hat{y}_j^t) \neq y_j^t\}$, and similarly the set of instances whose hinge-losses are greater than zero by $\Gamma^t = \{j \,|\, [1 - y_j^t \hat{y}_j^t]_+ > 0\}$.

## 3 Derived Problems

In this section we further explore the motivation for our problem setting by describing two different complex decision tasks and showing how they can be cast as special cases of our setting. We also would like to note that our approach can be employed in other prediction problems (see Sec. 7).

**Multilabel Categorization** In the multilabel categorization task each instance is associated with a set of relevant labels from the set $[k]$. The multilabel categorization task can be cast as a special case of a ranking task in which the goal is to rank the relevant labels above the irrelevant ones. Many learning algorithms for this task employ class-dependant features (for example, see [7]). For simplicity, assume that each class is associated with $n$ features and denote by $\phi(\mathbf{x}, r)$ the feature vector for class $r$. We would like to note that features obtained for different classes typically relay different information and are often substantially different.

A categorizer, or label ranker, is based on a weight vector $\boldsymbol{\omega}$. A vector $\boldsymbol{\omega}$ induces a score for each class $\boldsymbol{\omega} \cdot \phi(\mathbf{x}, r)$ which, in turn, defines an ordering of the classes. A learner is required to build a vector $\boldsymbol{\omega}$ that successfully ranks the labels according to their relevance, namely for each pair of classes $(r, s)$ such that $r$ is relevant while $s$ is not, the class $r$ should be ranked higher than the class $s$. Thus we require that $\boldsymbol{\omega} \cdot \phi(\mathbf{x}, r) > \boldsymbol{\omega} \cdot \phi(\mathbf{x}, s)$ for every such pair $(r, s)$. We say that a label ranking is imperfect if there exists *any* pair $(r, s)$ which violates this requirement. The loss associated with each such violation is $[1 - (\boldsymbol{\omega} \cdot \phi(\mathbf{x}, r) - \boldsymbol{\omega} \cdot \phi(\mathbf{x}, s))]_+$ and the loss of the categorizer is defined as the maximum over the losses induced by the violated pairs. In order to map the problem to our setting, we define a virtual instance for every pair $(r, s)$ such that $r$ is relevant and $s$ is not. The new instance is the $n$ dimensional vector defined by $\phi(\mathbf{x}, r) - \phi(\mathbf{x}, s)$. The label associated with all of the instances is set to 1. It is clear that an imperfect categorizer makes a prediction mistake on at least one of the instances, and that the losses defined by both problems are the same.

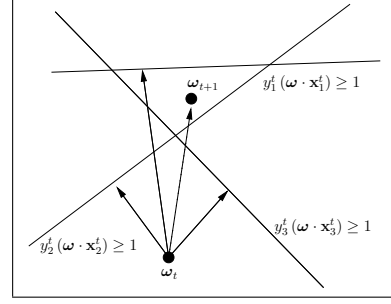

Figure 1: Illustration of the simultaneous projections algorithm: each instance casts a constraint on $\boldsymbol{\omega}$ and each such constraint defines a halfspace of feasible solutions. We project on each halfspace in parallel and the new vector is a weighted average of these projections

**Ordinal Regression** In the problem of ordinal regression an instance $\mathbf{x}$ is a vector of $n$ features that is associated with a target rank $y \in [k]$. A learning algorithm is required to find a vector $\boldsymbol{\omega}$ and $k$ thresholds $b_1 \leq \cdots \leq b_{k-1} \leq b_k = \infty$. The value of $\boldsymbol{\omega} \cdot \mathbf{x}$ provides a score from which the prediction value can be defined as the smallest index $i$ for which $\boldsymbol{\omega} \cdot \mathbf{x} < b_i$, $\hat{y} = \min\{i | \boldsymbol{\omega} \cdot \mathbf{x} < b_i\}$. In order to obtain a correct prediction, an ordinal regressor is required to ensure that $\boldsymbol{\omega} \cdot \mathbf{x} \geq b_i$ for all $i < y$ and that $\boldsymbol{\omega} \cdot \mathbf{x} < b_i$ for $i \geq y$. It is considered a prediction mistake if any of these constraints is violated. In order to map the ordinal regression task to our setting, we introduce $k - 1$ instances. Each instance is a vector in $\mathbb{R}^{n+k-1}$. The first $n$ entries of the vector are set to be the elements of $\mathbf{x}$, the remaining $k - 1$ entries are set to $-\delta_{i,j}$. That is, the $i$'th entry in the $j$'th vector is set to $-1$ if $i = j$ and to 0 otherwise. The label of the first $y - 1$ instances is 1, while the remaining $k - y$ instances are labeled as $-1$. Once we learned an expanded vector in $\mathbb{R}^{n+k-1}$, the regressor $\boldsymbol{\omega}$ is obtained by taking the first $n$ components of the expanded vector and the thresholds $b_1, \ldots, b_{k-1}$ are set to be the last $k - 1$ elements. A prediction mistake of any of the instances corresponds to an incorrect rank in the original problem.

## 4  Simultaneous Projection Algorithms

Recall that on trial $t$ the algorithm receives a matrix, $\mathbf{X}^t$, of $k_t$ instances, and predicts $\hat{\mathbf{y}}^t = \mathbf{X}^t \boldsymbol{\omega}^t$. After performing its prediction, the algorithm receives the corresponding labels $\mathbf{y}^t$. Each such instance-label pair casts a constraint on $\boldsymbol{\omega}_t$, $y_j^t \left( \boldsymbol{\omega}^t \cdot \mathbf{x}_j^t \right) \geq 1$. If all the constraints are satisfied by $\boldsymbol{\omega}^t$ then $\boldsymbol{\omega}^{t+1}$ is set to be $\boldsymbol{\omega}^t$ and the algorithm proceeds to the next trial. Otherwise, we would like to set $\boldsymbol{\omega}^{t+1}$ as close as possible to $\boldsymbol{\omega}^t$ while satisfying all constraints.

Such an aggressive approach may be sensitive to outliers and over-fitting. Thus, we allow some of the constraints to remain violated by introducing a tradeoff between the change to $\boldsymbol{\omega}^t$ and the loss attained on $(\mathbf{X}^t, \mathbf{y}^t)$. Formally, we would like to set $\boldsymbol{\omega}^{t+1}$ to be the solution of the following optimization problem, $\min_{\boldsymbol{\omega} \in \mathbb{R}^n} \frac{1}{2} \| \boldsymbol{\omega} - \boldsymbol{\omega}^t \|^2 + C \ell(\boldsymbol{\omega}; (\mathbf{X}^t, \mathbf{y}^t))$, where $C$ is a tradeoff parameter. As we discuss below, this formalism effectively translates to a cap on the maximal change to $\boldsymbol{\omega}^t$. We rewrite the above optimization by introducing a *single* slack variable as follows:

$$\min_{\boldsymbol{\omega} \in \mathbb{R}^n, \xi \geq 0} \frac{1}{2} \left\| \boldsymbol{\omega} - \boldsymbol{\omega}^t \right\|^2 + C\xi \quad \text{s.t.} \quad \forall j \in [k_t] : y_j^t \left( \boldsymbol{\omega} \cdot \mathbf{x}_j^t \right) \geq 1 - \xi \ , \ \xi \geq 0 \ . \quad (1)$$

We denote the objective function of Eq. (1) by $\mathcal{P}^t$ and refer to it as the *instantaneous* primal problem to be solved on trial $t$. The dual optimization problem of $\mathcal{P}^t$ is the maximization problem

$$\max_{\alpha_1^t, \ldots, \alpha_{k_t}^t} \sum_{j=1}^{k_t} \alpha_j^t - \frac{1}{2} \left\| \boldsymbol{\omega}^t + \sum_{j=1}^{k_t} \alpha_j^t y_j^t \mathbf{x}_j^t \right\|^2 \quad \text{s.t.} \quad \sum_{j=1}^{k_t} \alpha_j^t \leq C \ , \ \forall j : \alpha_j^t \geq 0 \quad . \quad (2)$$

Each dual variable corresponds to a single constraint of the primal problem. The minimizer of the primal problem is calculated from the optimal dual solution as follows, $\boldsymbol{\omega}^{t+1} = \boldsymbol{\omega}^t + \sum_{j=1}^{k_t} \alpha_j^t y_j^t \mathbf{x}_j^t$.

Unfortunately, in the common case, where each $\mathbf{x}_j^t$ is in an arbitrary orientation, there does not exist an analytic solution for the dual problem (Eq. (2)). We tackle the problem by breaking it down into $k_t$ reduced problems, each of which focuses on a single dual variable. Formally, for the $j$'th variable, the $j$'th reduced problem solves Eq. (2) while fixing $\alpha_{j'}^t = 0$ for all $j' \neq j$. Each reduced optimization problem amounts to the following problem

$$\max_{\alpha_j^t} \; \alpha_j^t - \frac{1}{2} \left\| \boldsymbol{\omega}^t + \alpha_j^t y_j^t \mathbf{x}_j^t \right\|^2 \quad \text{s.t.} \quad \alpha_j^t \in [0, C] \; . \tag{3}$$

We next obtain an exact or approximate solution for each reduced problem as if it were independent of the rest. We then choose a distribution $\boldsymbol{\mu}^t \in \Delta_{k_t}$, where $\Delta_{k_t} = \{ \boldsymbol{\mu} \in \mathbb{R}^{k_t} : \sum_j \mu_j = 1, \; \mu_j \geq 0 \}$ is the probability simplex, and multiply each $\alpha_j^t$ by the corresponding $\mu_j^t$. Since $\boldsymbol{\mu}^t \in \Delta_{k_t}$, this yields a feasible solution to the dual problem defined in Eq. (2) for the following reason. Each $\mu_j^t \alpha_j^t \geq 0$ and the fact that $\alpha_j^t \leq C$ implies that $\sum_{j=1}^{k_t} \mu_j^t \alpha_j^t \leq C$. Finally, the algorithm uses the combined solution and sets $\boldsymbol{\omega}^{t+1} = \boldsymbol{\omega}^t + \sum_{j=1}^{k_t} \mu_j^t \alpha_j^t y_j^t \mathbf{x}_j^t$.

We next present three schemes to obtain a solution for the reduced problem (Eq. (3)) and then combine the solution into a single update.

**Simultaneous Perceptron:** The simplest of the update forms generalizes the famous Perceptron algorithm from [8] by setting $\alpha_j^t$ to $C$ if the $j$'th instance is incorrectly labeled, and to 0 otherwise. We similarly set the weight $\mu_j^t$ to be $\frac{1}{|\mathcal{M}^t|}$ for $j \in \mathcal{M}^t$ and to 0 otherwise. We abbreviate this scheme as the *SimPerc* algorithm.

**Soft Simultaneous Projections:** The soft simultaneous projections scheme uses the fact that each reduced problem has an analytic solution, yielding $\alpha_j^t = \min \left\{ C, \ell \left( \boldsymbol{\omega}^t; (\mathbf{x}_j^t, y_j^t) \right) / \left\| \mathbf{x}_j^t \right\|^2 \right\}$. We independently assign each $\alpha_j^t$ this optimal solu-

> Input:
>     Aggressiveness parameter $C > 0$
> Initialize:
>     $\boldsymbol{\omega}_1 = (0, \ldots, 0)$
> For $t = 1, 2, \ldots, T$:
>     Receive instance matrix $X^t \in \mathbb{R}^{k_t \times n}$
>     Predict $\hat{\mathbf{y}}^t = \mathbf{X}^t \boldsymbol{\omega}^t$
>     Receive correct labels $\mathbf{y}^t$
>     Suffer loss $\ell \left( \boldsymbol{\omega}^t; (\mathbf{X}^t, \mathbf{y}^t) \right)$
>     If $\ell > 0$:
>         Choose importance weights $\boldsymbol{\mu}^t \in \Delta_{k_t}$
>         Choose individual dual solutions $\alpha_j^t$
>         Update $\boldsymbol{\omega}^{t+1} = \boldsymbol{\omega}^t + \sum_{j=1}^{k_t} \mu_j^t \alpha_j^t y_j^t \mathbf{x}_j^t$

Figure 2: Simultaneous projections algorithm.

tion. We next set $\mu_j^t$ to be $\frac{1}{|\Gamma^t|}$ for $j \in \Gamma^t$ and to 0 otherwise. We would like to comment that this solution may update $\alpha_j^t$ also for instances which were correctly classified as long as the margin they attain is not sufficiently large. We abbreviate this scheme as the *SimProj* algorithm.

**Conservative Simultaneous Projections:** Combining ideas from both methods, the conservative simultaneous projections scheme optimally sets $\alpha_j^t$ according to the analytic solution. The difference with the SimProj algorithm lies in the selection of $\boldsymbol{\mu}^t$. In the conservative scheme only the instances which were incorrectly predicted ($j \in \mathcal{M}^t$) are assigned a positive weight. Put differently, $\mu_j^t$ is set to $\frac{1}{|\mathcal{M}^t|}$ for $j \in \mathcal{M}^t$ and to 0 otherwise. We abbreviate this scheme as the *ConProj* algorithm.

To recap, on each trial $t$ we obtain a feasible solution for the instantaneous dual given in Eq. (2). This solution combines independently calculated $\alpha_j^t$, according to a weight vector $\boldsymbol{\mu}^t \in \Delta_{k_t}$. While this solution may not be optimal, it does constitutes an infrastructure for obtaining a mistake bound and, as we demonstrate in Sec. 6, performs well in practice.

## 5   Analysis

The algorithms described in the previous section perform updates in order to increase the instantaneous dual problem defined in Eq. (2). We now use the mistake bound model to derive an upper bound on the number of trials on which the predictions of SimPerc and ConProj algorithms are imperfect. Following [6], the first step in the analysis is to tie the instantaneous dual problems to

a global loss function. To do so, we introduce a primal optimization problem defined over the *entire* sequence of examples as follows, $\min_{\boldsymbol{\omega} \in \mathbb{R}^n} \frac{1}{2} \|\boldsymbol{\omega}\|^2 + C \sum_{t=1}^{T} \ell\left(\boldsymbol{\omega}; (X^t, Y^t)\right)$ . We rewrite the optimization problem as the following equivalent constrained optimization problem,

$$\min_{\boldsymbol{\omega} \in \mathbb{R}^n, \boldsymbol{\xi} \in \mathbb{R}^T} \frac{1}{2} \|\boldsymbol{\omega}\|^2 + C \sum_{t=1}^{T} \xi_t \quad \text{s.t.} \quad \forall t \in [T], \forall j \in [k_t] : y_j^t \left(\boldsymbol{\omega} \cdot \mathbf{x}_j^t\right) \geq 1 - \xi_t \quad \forall t : \xi_t \geq 0. \quad (4)$$

We denote the value of the objective function at $(\boldsymbol{\omega}, \boldsymbol{\xi})$ for this optimization problem by $\mathcal{P}(\boldsymbol{\omega}, \boldsymbol{\xi})$. A competitor who may see the entire sequence of examples in advance may in particular set $(\boldsymbol{\omega}, \boldsymbol{\xi})$ to be the minimizer of the problem which we denote by $(\boldsymbol{\omega}^\star, \boldsymbol{\xi}^\star)$. Standard usage of Lagrange multipliers yields that the dual of Eq. (4) is,

$$\max_{\boldsymbol{\lambda}} \sum_{t=1}^{T} \sum_{j=1}^{k_t} \lambda_{t,j} - \frac{1}{2} \left\| \sum_{t=0}^{T} \sum_{j=1}^{k_t} \lambda_{t,j} \, y_j^t \, \mathbf{x}_j^t \right\|^2 \quad \text{s.t.} \quad \forall t : \sum_{j=1}^{k_t} \lambda_{t,j} \leq C \quad \forall t, j : \lambda_{t,j} \geq 0 \; . \quad (5)$$

We denote the value of the objective function of Eq. (5) by $\mathcal{D}(\boldsymbol{\lambda}_1, \cdots, \boldsymbol{\lambda}_T)$, where each $\boldsymbol{\lambda}_t$ is a vector in $\mathbb{R}^{k_t}$. Through our derivation we use the fact that any set of dual variables $\boldsymbol{\lambda}_1, \cdots, \boldsymbol{\lambda}_T$ defines a feasible solution $\boldsymbol{\omega} = \sum_{t=1}^{T} \sum_{j=1}^{k_t} \lambda_{t,j} y_j^t \mathbf{x}_j^t$ with a corresponding assignment of the slack variables.

Clearly, the optimization problem given by Eq. (5) depends on all the examples from the first trial through time step $T$ and thus can only be solved in hindsight. We note however, that if we ensure that $\lambda_{s,j} = 0$ for all $s > t$ then the dual function no longer depends on instances occurring on rounds proceeding round $t$. As we show next, we use this primal-dual view to derive the skeleton algorithm from Fig. 2 by finding a new feasible solution for the dual problem on every trial. Formally, the instantaneous dual problem, given by Eq. (2), is equivalent (after omitting an additive constant) to the following constrained optimization problem,

$$\max_{\boldsymbol{\lambda}} \mathcal{D}(\boldsymbol{\lambda}_1, \cdots, \boldsymbol{\lambda}_{t-1}, \boldsymbol{\lambda}, \mathbf{0}, \cdots, \mathbf{0}) \quad \text{s.t.} \quad \boldsymbol{\lambda} \geq \mathbf{0} \; , \quad \sum_{j=1}^{k_t} \lambda_j \leq C \; . \quad (6)$$

That is, the instantaneous dual problem is obtained from $\mathcal{D}(\boldsymbol{\lambda}_1, \cdots, \boldsymbol{\lambda}_T)$ by fixing $\boldsymbol{\lambda}_1, \ldots, \boldsymbol{\lambda}_{t-1}$ to the values set in previous rounds, forcing $\boldsymbol{\lambda}_{t+1}$ through $\boldsymbol{\lambda}_T$ to the zero vectors, and choosing a feasible vector for $\boldsymbol{\lambda}_t$. Given the set of dual variables $\boldsymbol{\lambda}_1, \ldots, \boldsymbol{\lambda}_{t-1}$ it is straightforward to show that the prediction vector used on trial $t$ is $\boldsymbol{\omega}^t = \sum_{s=1}^{t-1} \sum_j \lambda_{s,j} y_j^s \mathbf{x}_j^s$. Equipped with these relations and omitting constants which do not depend on $\boldsymbol{\lambda}_t$ Eq. (6) can be rewritten as,

$$\max_{\lambda_1, \ldots, \lambda_{k_t}} \sum_{j=1}^{k_t} \lambda_j - \frac{1}{2} \left\| \boldsymbol{\omega}^t + \sum_{j=1}^{k_t} \lambda_j y_j^t \mathbf{x}_j^t \right\|^2 \quad \text{s.t.} \quad \forall j : \lambda_j \geq 0, \quad \sum_{j=1}^{k_t} \lambda_j \leq C \; . \quad (7)$$

The problems defined by Eq. (7) and Eq. (2) are equivalent. Thus, weighing the variables $\alpha_1^t, \ldots, \alpha_{k_t}^t$ by $\mu_1^t, \ldots, \mu_{k_t}^t$ also yields a feasible solution for the problem defined in Eq. (6), namely $\lambda_{t,j} = \mu_j^t \alpha_j^t$. We now tie all of these observations together by using the weak-duality theorem. Our first bound is given for the SimPerc algorithm.

**Theorem 1.** *Let $\left(\mathbf{X}^1, \mathbf{y}^1\right), \ldots, \left(\mathbf{X}^T, \mathbf{y}^T\right)$ be a sequence of examples where $\mathbf{X}^t$ is a matrix of $k_t$ examples and $\mathbf{y}^t$ are the associated labels. Assume that for all $t$ and $j$ the norm of an instance $\mathbf{x}_j^t$ is at most $R$. Then, for any $\boldsymbol{\omega}^\star \in \mathbb{R}^n$ the number of trials on which the prediction of SimPerc is imperfect is at most,*

$$\frac{\frac{1}{2} \|\boldsymbol{\omega}^\star\|^2 + C \sum_{t=1}^{T} \ell\left(\boldsymbol{\omega}^\star; (\mathbf{X}^t, \mathbf{y}^t)\right)}{C - \frac{1}{2} C^2 R^2} \; .$$

*Proof.* To prove the theorem we make use of the weak-duality theorem. Recall that any dual feasible solution induces a value for the dual's objective function which is upper bounded by the optimum value of the primal problem, $\mathcal{P}(\boldsymbol{\omega}^\star, \boldsymbol{\xi}^\star)$. In particular, the solution obtained at the end of trial $T$ is dual feasible, and thus $\mathcal{D}(\boldsymbol{\lambda}_1, \ldots, \boldsymbol{\lambda}_T) \leq \mathcal{P}(\boldsymbol{\omega}^\star, \boldsymbol{\xi}^\star)$ . We now rewrite the left hand-side of the above equation as the following sum,

$$\mathcal{D}(\mathbf{0}, \ldots, \mathbf{0}) + \sum_{t=1}^{T} \left[ \mathcal{D}(\boldsymbol{\lambda}_1, \ldots, \boldsymbol{\lambda}_t, \mathbf{0}, \ldots, \mathbf{0}) - \mathcal{D}(\boldsymbol{\lambda}_1, \ldots, \boldsymbol{\lambda}_{t-1}, \mathbf{0}, \ldots, \mathbf{0}) \right] \; . \quad (8)$$

Note that $\mathcal{D}(\mathbf{0}, \ldots, \mathbf{0})$ equals $0$. Therefore, denoting by $\Delta_t$ the difference in two consecutive dual objective values, $\mathcal{D}(\boldsymbol{\lambda}_1, \ldots, \boldsymbol{\lambda}_t, \mathbf{0}, \ldots, \mathbf{0}) - \mathcal{D}(\boldsymbol{\lambda}_1, \ldots, \boldsymbol{\lambda}_{t-1}, \mathbf{0}, \ldots, \mathbf{0})$, we get that $\sum_{t=1}^{T} \Delta_t \leq \mathcal{P}(\boldsymbol{\omega}^\star, \boldsymbol{\xi}^\star)$. We now turn to bounding $\Delta_t$ from below. First, note that if the prediction on trial $t$ is perfect ($\mathcal{M}^t = \emptyset$) then SimPerc sets $\boldsymbol{\lambda}_t$ to the zero vector and thus $\Delta_t = 0$. We can thus focus on trials for which the algorithm's prediction is imperfect. We remind the reader that by unraveling the update of $\boldsymbol{\omega}^t$ we get that $\boldsymbol{\omega}^t = \sum_{s<t} \sum_{j=1}^{k_s} \lambda_{s,j} y_j^s \mathbf{x}_j^s$. We now rewrite $\Delta_t$ as follows,

$$\Delta_t = \sum_{j=1}^{k_t} \lambda_{t,j} - \frac{1}{2} \left\| \boldsymbol{\omega}^t + \sum_{j=1}^{k_t} \lambda_{t,j} y_j^t \mathbf{x}_j^t \right\|^2 + \frac{1}{2} \left\| \boldsymbol{\omega}^t \right\|^2 \quad . \tag{9}$$

By construction, $\lambda_{t,j} = \mu_j^t \alpha_j^t$ and $\sum_{j=1}^{k_t} \mu_j^t = 1$, which lets us further expand Eq. (9) and write,

$$\Delta_t = \sum_{j=1}^{k_t} \mu_j^t \alpha_j^t - \frac{1}{2} \left\| \boldsymbol{\omega}^t + \sum_{j=1}^{k_t} \mu_j^t \alpha_j^t y_j^t \mathbf{x}_j^t \right\|^2 + \frac{1}{2} \sum_{j=1}^{k_t} \mu_j^t \left\| \boldsymbol{\omega}^t \right\|^2 \quad .$$

The squared norm, $\|\cdot\|^2$ is a convex function in its vector argument and thus $\Delta_t$ is concave, which yields the following lower bound on $\Delta_t$,

$$\Delta_t \geq \sum_{j=1}^{k_t} \mu_j^t \left[ \alpha_j^t - \frac{1}{2} \left\| \boldsymbol{\omega}^t + \alpha_j^t y_j^t \mathbf{x}_j^t \right\|^2 + \frac{1}{2} \left\| \boldsymbol{\omega}^t \right\|^2 \right] \quad . \tag{10}$$

The SimPerc algorithm sets $\mu_j^t$ to be $1/|\mathcal{M}^t|$ for all $j \in \mathcal{M}^t$ and to be $0$ otherwise. Furthermore, for all $j \in \mathcal{M}^t$, $\alpha_j^t$ is set to $C$. Thus, the right hand-side of Eq. (10) can be further simplified and written as,

$$\Delta_t \geq \sum_{j \in \mathcal{M}^t} \mu_j^t \left[ C - \frac{1}{2} \left\| \boldsymbol{\omega}^t + C y_j^t \mathbf{x}_j^t \right\|^2 + \frac{1}{2} \left\| \boldsymbol{\omega}^t \right\|^2 \right] \quad .$$

We expand the norm in the above equation and obtain that,

$$\Delta_t \geq \sum_{j \in \mathcal{M}^t} \mu_j^t \left[ C - \frac{1}{2} \left\| \boldsymbol{\omega}^t \right\|^2 - C y_j^t \boldsymbol{\omega}^t \cdot \mathbf{x}_j^t - \frac{1}{2} C^2 \left\| y_j^t \mathbf{x}_j^t \right\|^2 + \frac{1}{2} \left\| \boldsymbol{\omega}^t \right\|^2 \right] \quad . \tag{11}$$

The set $\mathcal{M}^t$ consists of indices of instances which were incorrectly classified. Thus, $y_j^t(\boldsymbol{\omega}^t \cdot \mathbf{x}_j^t) \leq 0$ for every $j \in \mathcal{M}^t$. Therefore, $\Delta_t$ can further be bounded from below as follows,

$$\Delta_t \geq \sum_{j \in \mathcal{M}^t} \mu_j^t \left[ C - \frac{1}{2} C^2 \left\| y_j^t \mathbf{x}_j^t \right\|^2 \right] \geq \sum_{j \in \mathcal{M}^t} \mu_j^t \left[ C - \frac{1}{2} C^2 R^2 \right] = C - \frac{1}{2} C^2 R^2 \quad , \tag{12}$$

where for the second inequality we used the fact that the norm of all the instances is bounded by $R$. To recap, we have shown that on trials for which the prediction is imperfect $\Delta_t \geq C - \frac{1}{2} C^2 R^2$, while in perfect trials where no mistake is made $\Delta_t = 0$. Putting all the inequalities together we obtain the following bound,

$$\left( C - \frac{1}{2} C^2 R^2 \right) \epsilon \leq \sum_{t=1}^{T} \Delta_t = \mathcal{D}(\boldsymbol{\lambda}_1, \ldots, \boldsymbol{\lambda}_T) \leq \mathcal{P}(\boldsymbol{\omega}^\star, \boldsymbol{\xi}^\star) \quad , \tag{13}$$

where $\epsilon$ is the number of imperfect trials. Finally, rewriting $\mathcal{P}(\boldsymbol{\omega}^\star, \boldsymbol{\xi}^\star)$ as $\frac{1}{2} \|\boldsymbol{\omega}^\star\|^2 + C \sum_{t=1}^{T} \ell(\boldsymbol{\omega}^\star; (\mathbf{X}^t, \mathbf{y}^t))$ yields the bound stated in the theorem. $\square$

The ConProj algorithm updates the same set of dual variables as the SimPerc algorithm, but selects $\alpha_j^t$ to be the optimal solution of Eq. (3). Thus, the value of $\Delta_t$ attained by the ConProj algorithm is never lower than the value attained by the SimPerc algorithm. The following corollary is a direct consequence of this observation.

**Corollary 1.** *Under the same conditions of Thm. 1 and for any $\boldsymbol{\omega}^\star \in \mathbb{R}^n$, the number of trials on which the prediction of ConProj is imperfect is at most,*

$$\frac{\frac{1}{2} \|\boldsymbol{\omega}^\star\|^2 + C \sum_{t=1}^{T} \ell(\boldsymbol{\omega}^\star; (\mathbf{X}^t, \mathbf{y}^t))}{C - \frac{1}{2} C^2 R^2} \quad .$$

| username | k | m | SimProj | ConProj | SimPerc | Max-SP | Max-MP | Mira |
|---|---|---|---|---|---|---|---|---|
| beck-s | 101 | 1973 | 50.0 | 55.2 | 55.9 | 56.6 | 63.8 | 63.7 |
| farmer-d | 25 | 3674 | 27.4 | 30.3 | 30.7 | 30.0 | 28.6 | 31.8 |
| kaminski-v | 41 | 4479 | 43.1 | 47.8 | 47.0 | 49.5 | 49.6 | 47.3 |
| kitchen-l | 47 | 4017 | 42.9 | 47.0 | 49.0 | 48.0 | 54.9 | 52.6 |
| lokay-m | 11 | 2491 | 18.8 | 25.3 | 25.3 | 23.0 | 25.4 | 25.3 |
| sanders-r | 30 | 1190 | 20.7 | 25.6 | 23.2 | 23.8 | 36.3 | 34.1 |
| williams-w3 | 18 | 2771 | 4.2 | 5.0 | 5.4 | 4.2 | 5.8 | 5.9 |

Table 1: The percentage of online mistakes of the three variants compared to Max-Update (Single prototype (SP) and Multi prototype (MP)) and the Mira algorithm. Experiments were performed on seven users of the Enron data set.

Note that the predictions of the SimPerc algorithm do not depend on the specific value of $C$, thus for $R = 1$ and an optimal choice of $C$ the bound attained in Thm. 1 now becomes.

$$\ell\left(\boldsymbol{\omega}^{\star};(\mathbf{X}^t,\mathbf{y}^t)\right) + \frac{1}{2}\|\boldsymbol{\omega}^{\star}\|^2 + \frac{1}{2}\sqrt{\|\boldsymbol{\omega}^{\star}\|^4 + \|\boldsymbol{\omega}^{\star}\|^2\ell\left(\boldsymbol{\omega}^{\star};(\mathbf{X}^t,\mathbf{y}^t)\right)} \ .$$

We omit the proof for lack of space, see [6] for a closely related analysis.

We conclude this section with a few closing words about the SimProj variant. The SimPerc and ConProj algorithms ensure a minimal increase in the dual by focusing solely on classification errors and ignoring margin errors. While this approach ensures a sufficient increase of the dual, in practice it appears to be a double edged sword as the SimProj algorithm performs empirically better. This superior empirical performance can be motivated by a refined derivation of the optimal choice for $\boldsymbol{\mu}$. This derivation will be provided in a long version of this manuscript.

## 6 Experiments

In this section we describe experimental results in order to demonstrate some of the merits of our algorithms. We tested performance of the three variants described in Sec. 4 on a multiclass categorization task and compared them to previously studied algorithms for multiclass categorization. We compared our algorithms to the single-prototype and multi-prototype Max-Update algorithms from [9] and to the Mira algorithm [2]. The experiments were performed on the task of email classification using the Enron email dataset (Available at http://www.cs.cmu.edu/~enron/enron_mail_030204.tar.gz). The learning goal was to correctly classify email messages into user defined folders. Thus, the instances in this dataset are email messages, while the set of classes are the user defined folders denoted by $\{1, \ldots, k\}$. We ran the experiments on the sequence of email messages from 7 different users.

Since each user employs different criteria for email classification, we treated each person as a separate online learning problem. We represented each email message as a vector with a component for every word in the corpus. On each trial, and for each class $r$, we constructed class-dependent vectors as follows. We set $\phi_j(\mathbf{x}^t, r)$ to twice the number of time the $j$'th word appeared in the message if it had also appeared in a fifth of the messages previously assigned to folder $r$. Similarly, we set $\phi_j(\mathbf{x}^t, r)$ to minus the number of appearances of the word appeared if it had appeared in less than 2 percent of previous messages. In all other cases, we set $\phi_j(\mathbf{x}^t, r)$ to 0. This class-dependent construction is closely related to the construction given in [10]. Next, we employed the mapping described in Sec. 3, and defined a set of $k-1$ instances for each message as follows. Denote the relevant class by $r$, then for every irrelevant class $s \neq r$, we define an instance $\mathbf{x}_s^t = \phi(\mathbf{x}^t, r) - \phi(\mathbf{x}^t, s)$ and set its label to 1. All these instances were combined into a single matrix $\mathbf{X}^t$ and were provided to the algorithm in trial $t$.

The results of the experiments are summarized in Table 1. It is apparent that the SimProj algorithm outperforms all other algorithms. The performances of SimPerc and ConProj are comparable with no obvious winner. It is worth noting that the Mira algorithm finds the optimum of a projection problem on each trial while our algorithms only find an approximate solution. However, Mira employs a different approach in which there is a single input instance (instead of the set $\mathbf{X}^t$) and constructs multiple predictors (instead of a single vector $\boldsymbol{\omega}$). Thus, Mira employs a larger hypothesis space which is more difficult to learn in online settings. In addition, by employing a single vector

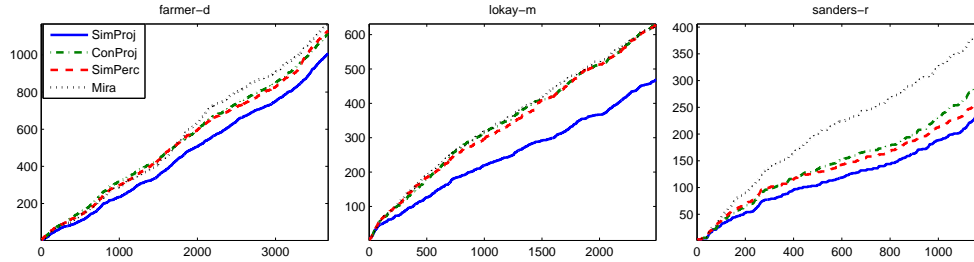

Figure 3: The cumulative number of mistakes as a function of the number of trials.

representation of the email message, Mira cannot benefit from feature selection which yields class-dependent features. It is also obvious that the simultaneous projection variants, while remaining simple to implement, consistently outperform the Max-Update technique which is commonly used in online multiclass classification. In Fig. 3 we plot the cumulative number of mistakes as a function of the trial number for 3 of the 7 users. The graphs clearly indicate the high correlation between the $SimPerc$ and $ConProj$ variants, while indicating the superiority of the $SimProj$ variant.

## 7 Extensions and discussion

We presented a new approach for online categorization with complex output structure. Our algorithms decouple the complex optimization task into multiple sub-tasks, each of which is simple enough to be solved analytically. While the dual representation of the online problem imposes a global constraint on *all* the dual variables, namely $\sum_j \alpha_j^t \leq C$, our framework of simultaneous projections which are followed by averaging the solutions automatically adheres with this constraint and hence constitute a feasible solution. It is worthwhile noting that our approach can also cope with *multiple* constraints of the more general form $\sum_j \nu_j \alpha_j \leq C$, where $\nu_j \geq 0$ for all $j$. The box constraint implied for each individual projection problem distils to $0 \leq \alpha_j \leq C/\nu_j$ and thus the simultaneous projection algorithm can be used verbatim. We are currently exploring the usage of this extension in complex decision problems with multiple structural constraints. Another possible extension is to replace the squared norm regularization with other twice differentiable penalty functions. Algorithms of this more general framework still attain similar mistake bounds and are easy to implement so long as the induced individual problems are efficiently solvable. A particularly interesting case is obtained when setting the penalty to the relative entropy. In this case we obtain a generalization of the Winnow and the EG algorithms [11, 12] for complex classification problems. Another interesting direction is the usage of simultaneous projections for problems with more constrained structured output such as max-margin networks [3].

## References

[1] J. Weston and C. Watkins. Support vector machines for multi-class pattern recognition. In *Proc. of the Seventh European Symposium on Artificial Neural Networks*, April 1999.

[2] K. Crammer and Y. Singer. Ultraconservative online algorithms for multiclass problems. *J. of Machine Learning Res.*, 3:951–991, 2003.

[3] B. Taskar, C. Guestrin, and D. Koller. Max-margin markov networks. In *Advances in Neural Information Processing Systems 17*, 2003.

[4] I. Tsochantaridis, T. Hofmann, T. Joachims, and Y. Altun. Support vector machine learning for interdependent and structured output spaces. In *Proc. of the 21st Intl. Conference on Machine Learning*, 2004.

[5] Yoav Freund and Robert E. Schapire. A decision-theoretic generalization of on-line learning and an application to boosting. *Journal of Computer and System Sciences*, 55(1):119–139, August 1997.

[6] S. Shalev-Shwartz and Y. Singer. Online learning meets optimization in the dual. In *Proc. of the Nineteenth Annual Conference on Computational Learning Theory*, 2006.

[7] R.E. Schapire and Y. Singer. BoosTexter: A boosting-based system for text categorization. *Machine Learning*, 32(2/3), 2000.

[8] F. Rosenblatt. The perceptron: A probabilistic model for information storage and organization in the brain. *Psychological Review*, 65:386–407, 1958. (Reprinted in *Neurocomputing* (MIT Press, 1988).).

[9] K. Crammer, O. Dekel, J. Keshet, S. Shalev-Shwartz, and Y. Singer. Online passive aggressive algorithms. *Journal of Machine Learning Research*, 7, Mar 2006.

[10] M. Fink, S. Shalev-Shwartz, Y. Singer, and S. Ullman. Online multiclass learning by interclass hypothesis sharing. In *Proc. of the 23rd International Conference on Machine Learning*, 2006.

[11] N. Littlestone. Learning when irrelevant attributes abound: A new linear-threshold algorithm. *Machine Learning*, 2:285–318, 1988.

[12] J. Kivinen and M. Warmuth. Exponentiated gradient versus gradient descent for linear predictors. *Information and Computation*, 132(1):1–64, January 1997.
